# PHASOR NEURAL NETWORKS

André J. Noest, N.I.B.R., NL-1105 AZ Amsterdam, The Netherlands.

## ABSTRACT

A novel network type is introduced which uses unit-length 2-vectors for local variables. As an example of its applications, associative memory nets are defined and their performance analyzed. Real systems corresponding to such 'phasor' models can be e.g. (neuro)biological networks of limit-cycle oscillators or optical resonators that have a hologram in their feedback path.

## INTRODUCTION

Most neural network models use either binary local variables or scalars combined with sigmoidal nonlinearities. Rather awkward coding schemes have to be invoked if one wants to maintain linear relations between the local signals being processed in e.g. associative memory networks, since the nonlinearities necessary for any nontrivial computation act directly on the range of values assumed by the local variables. In addition, there is the problem of representing signals that take values from a space with a different topology, e.g. that of the circle, sphere, torus, etc. Practical examples of such a signal are the orientations of edges or the directions of local optic flow in images, or the phase of a set of (sound or EM) waves as they arrive on an array of detectors. Apart from the fact that 'circular' signals occur in technical as well as biological systems, there are indications that some parts of the brain (e.g. olfactory bulb, cf. Dr.B.Baird's contribution to these proceedings) can use limit-cycle oscillators formed by local feedback circuits as functional building blocks, even for signals without circular symmetry. With respect to technical implementations, I had speculated before the conference whether it could be useful to code information in the phase of the beams of optical neurocomputers, avoiding slow optical switching elements and using only (saturating) optical amplification and a

hologram encoding the (complex) 'synaptic' weight factors. At the conference, I learnt that Prof. Dana Anderson had independently developed an optical device (cf. these proceedings) that basically works this way, at least in the slow-evolution limit of the dynamic hologram. Hopefully, some of the theory that I present here can be applied to his experiment. In turn, such implementations call for interesting extensions of the present models.

## BASIC ELEMENTS OF GENERAL PHASOR NETWORKS

Here I study the perhaps simplest non-scalar network by using unit-length 2-vectors (phasors) as continuous local variables. The signals processed by the network are represented in the relative phaseangles. Thus, the nonlinearities (unit-length 'clipping') act orthogonally to the range of the variables coding the information. The behavior of the network is invariant under any rigid rotation of the complete set of phasors, representing an arbitrary choice of a global reference phase. Statistical physicists will recognize the phasor model as a generalization of $O_2$-spin models to include vector-valued couplings.

All 2-vectors are treated algebraically as complex numbers, writing $|x|$ for the length, $/x/$ for the phase-angle, and $\bar{x}$ for the complex conjugate of a 2-vector x.

A phasor network then consists of $N \gg 1$ phasors $s_i$ , with $|s_i|=1$, interacting via couplings $c_{ij}$, with $c_{ii}= 0$. The $c_{ij}$ are allowed to be complex-valued quantities. For optical implementations this is clearly a natural choice, but it may seem less so for biological systems. However, if the coupling between two limitcycle oscillators with frequency f is mediated via a path having propagationdelay d, then that coupling in fact acquires a phaseshift of $f.d.2\pi$ radians. Thus, complex couplings can represent such systems more faithfully than the usual models which neglect propagationdelays altogether. Only 2-point couplings are treated here, but multi-point couplings $c_{ijk}$, etc., can be treated similarly.

The dynamics of each phasor depends only on its local field

$$h_i = \frac{1}{z} \sum_j c_{ij} s_j + n_i \quad , \quad \text{where z is the number of inputs}$$

$c_{ij} \neq 0$ per cell and $n_i$ is a local noise term (complex and Gaussian). Various dynamics are possible, and yield largely similar results: Continuous-time, parallel evolution: ("type A")

$$\frac{d}{dt}(/s_i/) = |h_i| \cdot \sin(/h_i/ - /s_i/)$$

Discrete-time updating: $s_i(t+\delta t) = h_i/|h_i|$, either serially in random i-sequence ("type B"), or in parallel for all i ("type C"). The natural time scale for type-B dynamics is obtained by scaling the discrete time-interval $\delta t$ as $\sim 1/N$; type-C dynamics has $\delta t = 1$.

### LYAPUNOV FUNCTION   (alias "ENERGY", or "HAMILTONIAN" )

If one limits the attention temporarily to purely deterministic ($n_i = 0$) models, then the question suggests itself whether a class of couplings exists for which one can easily find a Lyapunov function i.e. a function of the network variables that is monotonic under the dynamics. A well-known example[1] is the 'energy' of the binary and scalar Hopfield models with symmetric interactions. It turns out that a very similar function exists for phasor networks with type-A or B dynamics and a Hermitian matrix of couplings.

$$-H = \sum_i \bar{s}_i h_i = (1/z) \sum_{i,j} \bar{s}_i c_{ij} s_j$$

Hermiticity ($c_{ij} = \bar{c}_{ji}$) makes H real-valued and non-increasing in time. This can be shown as follows, e.g. for the serial dynamics (type B). Suppose, without loss of generality, that phasor i=1 is updated.

Then $-z H = z \bar{s}_1 h_1 + \sum_{i>1} \bar{s}_i c_{i1} s_1 + \sum_{i,j>1} \bar{s}_i c_{ij} s_j$

$= z \bar{s}_1 h_1 + s_1 \cdot \sum_{i>1} c_{i1} \bar{s}_i + \text{constant}.$

With Hermitian couplings, H becomes real-valued, and one also has

$$\sum_{i>1} c_{i1} \bar{s}_i = \sum_{i>1} \bar{c}_{1i} \bar{s}_i = z \bar{h}_1 .$$

Thus, $- H - \text{constant} = \bar{s}_1 h_1 + s_1 \bar{h}_1 = 2 \, \text{Re}(s_1 \bar{h}_1)$ .
Clearly, H is minimized with respect to $s_1$ by $s_1(t+1) = h_1/|h_1|$ .
Type-A dynamics has the same Lyapunovian, but type C is more complex. The existence of Hermitian interactions and the corresponding energy function simplifies greatly the understanding and design of phasor networks, although non-Hermitian networks can still have a Lyapunov-

function, and even networks for which such a function is not readily found can be useful, as will be illustrated later.

## AN APPLICATION : ASSOCIATIVE MEMORY.

A large class of collective computations, such as optimisations and content-addressable memory, can be realised with networks having an energy function. The basic idea is to define the relevant penalty function over the solution-space in the form of the generic 'energy' of the net, and simply let the network relax to minima of this energy. As a simple example, consider an associative memory built within the framework of Hermitian phasor networks.

In order to store a set of patterns in the network, i.e. to make a set of special states (at least approximatively) into attractive fixed points of the dynamics, one needs to choose an appropriate set of couplings. One particularly simple way of doing this is via the phasor-analog of "Hebb's rule" (note the Hermiticity)

$$c_{ij} = \sum_k^p s_i^{(k)} \cdot \bar{s}_j^{(k)}, \text{ where } s_i^{(k)} \text{ is phasor i in learned pattern k.}$$

The rule is understood to apply only to the input-sets $\partial i$ of each i. Such couplings should be realisable as holograms in optical networks, but they may seem unrealistic in the context of biological networks of oscillators since the phase-shift (e.g. corresponding to a delay) of a connection may not be changeable at will. However, the required coupling can still be implemented naturally if e.g. a few paths with different fixed delays exist between pairs of cells. The synaps in each path then simply becomes the projection of the complex coupling on the direction given by the phase of its path, i.e. it is just a classical Hebb-synapse that computes the correlation of its pre- and post-synaptic (imposed) signals, which now are phase-shifted versions of the phasors $s_i^{(k)}$. The required complex $c_{ij}$ are then realised as the vector sum over at least two signals arriving via distinct paths with corresponding phase-shift and real-valued synaps. Two paths suffice if they have orthogonal phase-shifts, but random phases will do as well if there are a reasonable number of paths.

We need to have a concise way of expressing how 'near' any state of the net is to one or more of the stored patterns. A natural way of doing this is via a set of p order parameters called "overlaps"

$$M_k = \frac{1}{N} \left| \sum_i^N s_i \cdot \bar{s}_i^{(k)} \right| \quad ; \quad 0 \leq M_k \leq 1 \; ; \; 1 \leq k \leq p .$$

Note the constraint on the p overlaps $\sum_k^p M_k^2 \leq 1$ if all the patterns
are orthogonal, or merely random in the limit $N \to \infty$. This will be
assumed from now on. Also, one sees at once that the whole behaviour
of the network does not depend on any rigid rotation of all phasors
over some angle since H, $M_k$, $c_{ij}$ and the dynamics are invariant under
multiplication of all $s_i$ by a fixed phasor : $s_i' = S \cdot s_i$ with $|S|=1$.

Let us find the performance at low loading: $N,p,z \to \infty$, with $p/z \to 0$
and zero local noise. Also assume an initial overlap m>0 with only
one pattern, say with k=1. Then the local field is

$$h_i = \frac{1}{z} \sum_{j \in \partial i} s_j \cdot \sum_k^p s_i^{(k)} \bar{s}_j^{(k)} = h_i^{(1)} + h_i^\star \quad , \text{ where }$$

$$h_i^{(1)} = \frac{1}{z} s_i^{(1)} \cdot \sum_{j \in \partial i} \bar{s}_j^{(1)} \cdot s_j = m_1 \cdot s_i^{(1)} \cdot S + 0(1/\sqrt{z}) \text{ , with } S \neq f(i); |S|=1,$$

and $$h_i^\star = \frac{1}{z} \sum_{k=2}^p s_i^{(k)} \cdot \sum_{j \in \partial i} \bar{s}_j^{(k)} \cdot s_j = 0(\sqrt{(p-1)/z}) .$$

Thus, perfect recall ($M_1=1$) occurs in one 'pass' at loadings $p/z \to 0$.

### EXACTLY SOLVABLE CASE: SPARSE and ASYMMETRIC couplings

Although it would be interesting to develop the full thermodynamics
of Hermitian phasor networks with p and z of order N (analogous to the
analysis of the finite-T Hopfield model by the teams of Amit[2] and van
Hemmen[3]), I will analyse here instead a model with sparse, asymmetric
connectivity, which has the great advantages of being exactly solvable
with relative ease, and of being arguably more realistic biologically
and more easily scalable technologically. In neurobiological networks
a cell has up to $z \cong 10^4$ asymmetric connections, whereas $N \cong 10^{11}$. This
probably has the same reason as applies to most VLSI chips, namely to
alleviate wiring problems. For my present purposes, the theoretical
advantage of getting some exact results is of primary interest[4].

Suppose each cell has z incoming connections from randomly selected
other cells. The state of each cell at time t depends on at most $z^t$
cells at time t=0. Thus, if $z^t \ll N^{1/2}$ and N large, then the respective

trees of 'ancestors' of any pair cells have no cells in common[4]. In particular, if $z \sim (\log N)^x$, for any finite x, then there are no common ancestors for any finite time t in the limit $N \to \infty$. For fundamental information-theoretic reasons, one can hope to be able to store p patterns with p at most of order z for any sort of 2-point couplings. Important questions to be settled are: What are the accuracy and speed of the recall process, and how large are the basins of the attractors representing recalled patterns?

Take again initial conditions (t=0) with, say, $m(t) = M_1 > M_{>1} = 0$. Allowing again local random Gaussian (complex) noise $n_i$, the local fields become, in now familiar notation, $h_i = h_i^{(1)} + h_i^* + n_i$. As in the previous section, the $h_i^{(1)}$ term consists of the 'signal' $m(t).s_i$ (modulo the rigid rotation S) and a random term of variance at most $1/z$. For $p \sim z$, the $h_i^*$ term becomes important. Being sums of $z(p-1)$ phasors oriented randomly relative to the signal, the $h_i^*$ are independent Gaussian zero-mean 2-vectors with variance $(p-1)/z$ , as p,z and $N \to \infty$ . Finally, let the local noises $n_i$ have variance $r^2$. Then the distribution of the $s_i(t+1)$ phasors can be found in terms of the signal m(t) and the total variance $a = (p/z) + r^2$ of the random $h_i^* + n_i$. After somewhat tedious algebraic manipulations (to be reported in detail elsewhere) one obtains the dynamic behaviour of m(t) :

$$m(t+1) = F(m(t),a) \qquad \text{for discrete parallel (type-C) dynamics,}$$
and
$$\frac{d}{dt} m(t) = F(m(t),a) - m(t) \qquad \text{for type-A or type-B dynamics ,}$$

where the function $F(m,a) =$

$$\frac{m}{4\sqrt{a\pi}} \int_{-\pi}^{+\pi} dx.(1+\cos 2x).\exp[-(m.\sin x)^2/a].(1+\text{erf}[(m.\cos x)/\sqrt{a}])$$

The attractive fixed points $M^*(a) = F(M^*,a)$ represent the retrieval accuracy when the loading-plus-noise factor equals a. See figure 1.

For $a \ll 1$ one obtains the expansion $1 - M^*(a) = a/4 + 3a^2/32 + O(a^3)$.

The recall solutions vanish continuously as $M^* \sim (a_c - a)^{1/2}$ at $a_c = \pi/4$.

One also obtains (at any t) the distribution of the phase scatter of the phasors around the ideal values occurring in the stored pattern.

$$P(/u_i/) = (1/2\pi).\exp(-m^2/a).(1+\sqrt{\pi}.L.\exp(L^2).(1+\mathrm{erf}(L))) ,$$

where $\quad L = (m/\sqrt{a}).\cos(/u_i/)$ , and $\quad u_i = s_i \bar{s}_i^{(k)}(\text{modulo } S)$.

Useful approximations for the high, respectively low M regimes are:

$$M \gg \sqrt{a} : \quad P(/u_i/) = (M/\sqrt{a\pi}).\exp[-(M./u_i/)^2/a] \quad ; \quad |/u_i/| \ll \pi/2$$

$$M \ll \sqrt{a} : \quad P(/u_i/) = (1/2\pi).(1+L.\sqrt{\pi})$$

Figure 1.

RETRIEVAL-ERROR and BASIN OF ATTRACTION versus LOADING + NOISE.

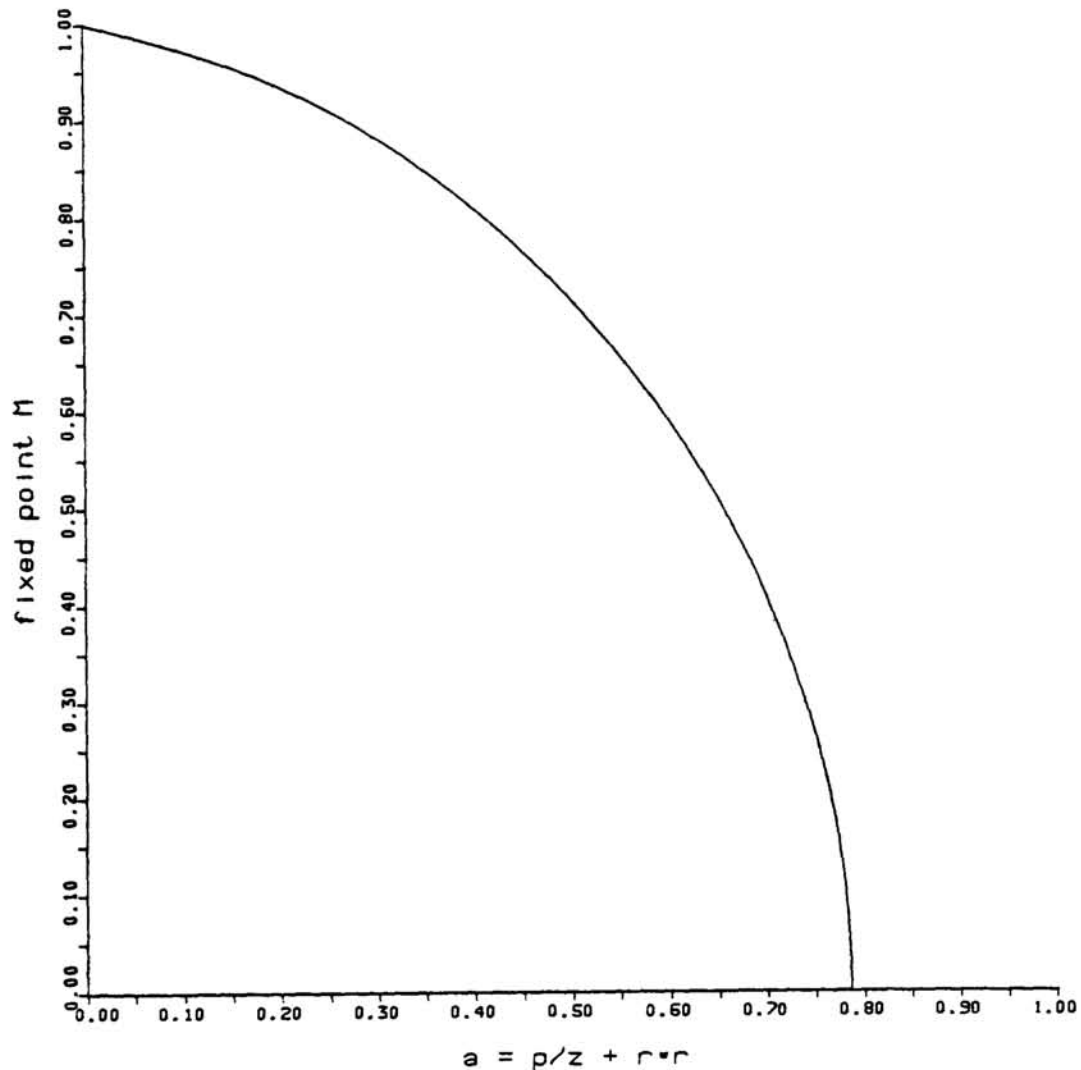

## DISCUSSION

It has been shown that the usual binary or scalar neural networks can be generalized to phasor networks, and that the general structure of the theoretical analysis for their use as associative memories can be extended accordingly. This suggests that many of the other useful applications of neural nets (back-prop, etc.) can also be generalized to a phasor setting. This may be of interest both from the point of view of solving problems naturally posed in such a setting, as well as from that of enabling a wider range of physical implementations, such as networks of limit-cycle oscillators, phase-encoded optics, or maybe even Josephson-junctions.

The performance of phasor networks turns out to be roughly similar to that of the scalar systems; the maximum capacity $p/z = \pi/4$ for phasor nets is slightly larger than its value $2/\pi$ for binary nets, but there is a seemingly faster growth of the recall error 1-M at small a (linear for phasors, against $\exp(-1/(2a))$ for binary nets). However, the latter measures cannot be compared directly since they stem from quite different order parameters. If one reduces recalled phasor patterns to binary information, performance is again similar. Finally, the present methods and results suggest several roads to further generalizations, some of which may be relevant with respect to natural or technical implementations. The first class of these involves local variables ranging over the k-sphere with k>1. The other generalizations involve breaking the O(n) (here n=2) symmetry of the system, either by forcing the variables to discrete positions on the circle (k-sphere), and/or by taking the interactions between two variables to be a more general function of the angular distance between them. Such models are now under development.

## REFERENCES

1. J.J.Hopfield, Proc.Nat.Acad.Sci.USA 79, 2554 (1982) and
   idem, Proc.Nat.Acad.Sci.USA 81, 3088 (1984).
2. D.J.Amit, H.Gutfreund and H.Sompolinski, Ann.Phys. 173, 30 (1987).
3. D.Grensing, R.Kuhn and J.L. van Hemmen, J.Phys.A 20, 2935 (1987).
4. B.Derrida, E.Gardner and A.Zippelius, Europhys.Lett. 4, 167 (1987)
